# ASSOCIATIVE LEARNING
# VIA INHIBITORY SEARCH

David H. Ackley
Bell Communications Research
Cognitive Science Research Group

## ABSTRACT

*ALVIS* is a reinforcement-based connectionist architecture that learns associative maps in continuous multidimensional environments. The discovered locations of positive and negative reinforcements are recorded in "do be" and "don't be" subnetworks, respectively. The outputs of the subnetworks relevant to the current goal are combined and compared with the current location to produce an error vector. This vector is backpropagated through a motor-perceptual mapping network to produce an action vector that leads the system towards do-be locations and away from don't-be locations. *ALVIS* is demonstrated with a simulated robot posed a target-seeking task.

## INTRODUCTION

The "backpropagation algorithm" or *generalized delta rule* (Rumelhart, Hinton, & Williams, 1986) is sometimes criticized on the grounds that it is a "supervised" learning algorithm, which requires a "teacher" to provide correct outputs, and apparently leaves open the question of how the teacher learned the right answers. However, work by Rumelhart (personal communication, 1987) and Miyata (1988) has shown how the environment that a system is embedded in can serve as the "teacher." If, as in this paper, a backpropagation network is posed the task of mapping from a vector $\theta$ of robot arm joint angles to the resulting vector $X$ of arm coordinates in space (the "forward kinematics problem"), then input-output training data can be obtained by supplying sets of joint angles to the arm and observing the resulting configurations.

Although this "environment as teacher" strategy shows how a "teacher" can come to possess useful information without an infinite regress learning it, it is not a complete solution. There are problems for which the "laws of physics" of an environment do not suffice to determine the solution. Suppose, for example, that a robot is posed the problem of learning to reach for different positions in space depending on which of a set of signals is currently presented, and that the only feedback available from the environment is success or failure information about the current arm configuration.

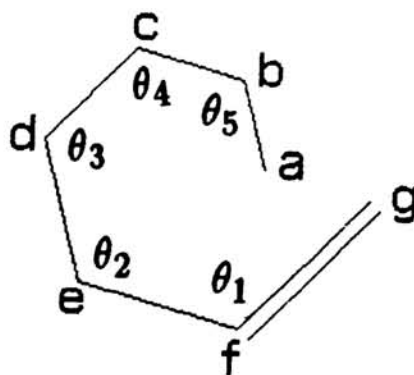

**Figure 1.** A "trunk" robot.

What is needed in such a case is a mechanism to search through the space of possible arm configurations, recording the successful configurations associated with the various inputs. *ALVIS* — Associative Learning Via Inhibitory Search — provides one such mechanism. The next section applies backpropagation to the $\theta \rightarrow X$ mapping and shows how the resulting network can sometimes be used to solve $X \rightarrow \theta$ problems. The third section, "Self-supervision and inhibitory search," integrates that network into the overall *ALVIS* algorithm. The final section contains some discussion and conclusions. An expanded version of this paper may be found in Ackley (1988).

## FORWARD AND INVERSE KINEMATICS

The *inverse kinematics* problem in controlling an arm is the problem of determining what joint angles are needed to produce a specific position and orientation of a hand. In the general case it is a difficult problem. An itch on your back suggests the kinds of questions that arise. Which hand should you use? Should you go up from around your waist, or down from over your shoulders? Can you be sure you know what will work without actually trying it?

From a computational standpoint, *forward kinematics* — deciding where your limbs will end up given a set of joint angles — is an easier problem.

Figure 1 depicts the planar "robot" that was used in this work. I call it the "trunk" robot. (The work discussed in Ackley (1988) also used a two-handed "pincer" robot.) Of course, the trunk is a far cry from a real robot, and the only significant constraint is that the possible joint angles are limited, but this suffices to pose non-trivial kinematics problems. The trunk has five joints, and each joint angle is limited to a range of $4°$ to $176°$ with respect to the previous limb.

I simulated a backpropagation network with five real-valued input units (the joint angles), sixty hidden units in a single layer, and twelve linear output units (the Cartesian joint positions). Joint angles were expressed in radians, so the range of input unit values was from about 0.07 to about 3.07. The configuration vector $X$ was represented by twelve output units corresponding to six pairs of $(x, y)$ coordi-

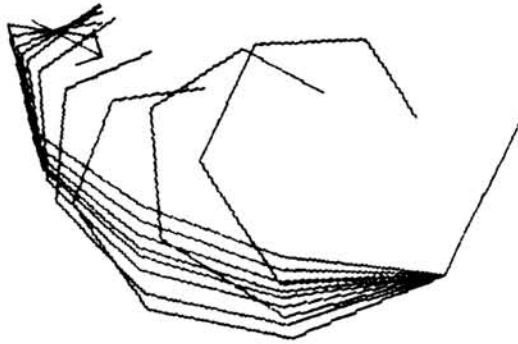

**Figure 2.** A "logarithmic strobe" display of the trunk's asymptotic convergence on a specified position and orientation. The arm position is displayed after iterations $1, 2, 4, 8, \ldots, 256$.

nates, one pair for each joint $a$ through $f$. With the trunk robot (though not with the "pincer") $f_x$ and $f_y$ have constant values, since they end up being part of the "anchor." The state of an output unit equals the sum of its inputs, and the error propagated out of an output unit equals the error propagated into it.

Errors were defined by the difference between the predicted configuration and the actual configuration, and after extensive training on the trunk robot forward kinematics problem, the network achieved high accuracy over most of the joint ranges.

In typical backpropagation applications, once the desired mapping has been learned, the backward "error channels" in the network are no longer used. However, suppose some other error computation, different from that used to train the weights, was then incorporated. Those errors can be propagated from the outputs of the trained network all the way back to the inputs. The goal is no longer to change weights in the network — since they already represent a useful mapping — but to use the trained network to *translate* output-space errors, however defined, into errors at the inputs to the network.

Figure 2 illustrates one use of this process, showing how a trained forward kinematics network can be used to perform a cheap kind of inverse kinematics. The figure shows superimposed outputs of the trained network under the influence of a task-specific error computation; in this case the trunk is trying to reduce the distances between the front and back of a "target arrow" and the front and back of its first arm section. The target arrow is defined by a head $(h_x, h_y)$ and a tail $(t_x, t_y)$. The errors for output units $a_x$ and $a_y$ are defined by $e(a_x) = h_x - a_x$ and $e(a_y) = h_y - a_y$, the errors for output units $b_x$ and $b_y$ are defined by $e(b_x) = t_x - b_x$ and $e(b_y) = t_y - b_y$, and the errors at all other outputs are set to zero.

The algorithm used to generate this behavior has the following steps:

1. Compute errors for one or more output units based on the current positions of the joints and the desired positioning and orienting information. If "close

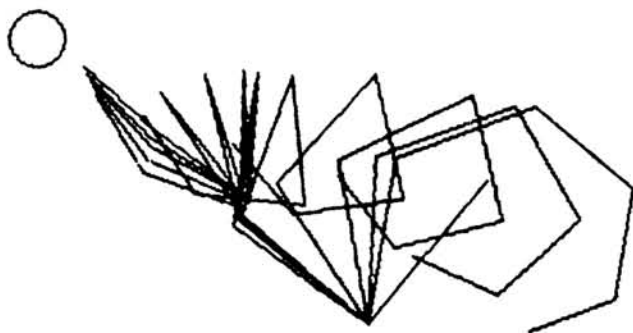

**Figure 3.** The trunk kinking itself.

enough" to the target, exit, otherwise, store these errors on the selected output units, and set the other error terms to zero.

2. Backpropagate the errors all the way through the network to the input units. This produces an error term $e(\theta_i)$ for each joint angle $\theta_i$. Produce new joint angles: $\theta_i' = \theta_i + ke(\theta_i)$ where $k$ is a scaling constant. Clip the joint angles against their minimum and maximum values.

3. Forward propagate through the network based on the new joint angles, to produce new current positions for the joints. Go to step 1.

Whereas the training phase has forward propagation of activations (states) followed by back propagation of errors, this usage reverses the order. Backpropagation of errors is followed by changes in inputs followed by forward propagation of activations. This is a general gradient descent technique usable when a backpropagation network can learn to map from a control space $\theta$ to an error or evaluation space $X$.

Figure 3 illustrates how gradient descent's familiar limitation can manifest itself: The target is reachable but the robot fails to reach it. The initial configuration was such that while approaching the target, the trunk kinked itself too short to reach. If the robot had "thought" to open $\theta_3$ instead of closing it, it could have succeeded. In that sense, the problem arises because the error computation only specified errors for the tip of the trunk, and not for the rest of the arm. If, instead, there were indications where *all* of the joints were to be placed, failures due to local minima could be greatly reduced.

## SELF-SUPERVISION AND INHIBITORY SEARCH

The feedback control network of the previous section locally minimizes joint position errors — however they are generated — by translating them into joint angle space and moving downhill. *ALVIS* uses the feedback control network for arm control; this section shows how *ALVIS* learns to generate appropriate joint position space errors given only a reinforcement signal. There are two key points. The first is this: Once an action producing a positive reinforcement has somehow been found,

the problem reduces to associative mapping between the input and the discovered correct output. In *ALVIS*, "do-be units" are used to record such successes. The second point is this: When negative reinforcement occurs, the current configuration can be associated with the input in a behavior-reversed fashion — as a place to *avoid* in the future. In *ALVIS*, "don't-be units" are used to record such failures.

The overall idea, then, is to perform *inhibitory search* by remembering failures as they occur and avoiding them in the future, and to perform associative learning by remembering successful configurations as the search process uncovers them and recreating them in the future. In effect, *ALVIS* constructs input-dependent "attractors" at arm configurations associated with success and "repellors" at configurations associated with failure. Figure 4 summarizes the algorithm. A few points to note are these:

- The do-be and don't-be units use the *spherical* non-linear function explored by Burr & Hanson (1987). The response of a spherical unit is maximal and equal to one when the input vector and the weight vector are identical. The response of the unit decreases monotonically with the Euclidean distance between the two vectors, and the radius $r$ governs the rate of decay.

- The don't-be units of each subnetwork (i.e., relevant to one goal) are in a competitive network (see, e.g., Feldman 1982). The don't-be unit with the largest activation value (which is a function of both the distance from the current position and the radius) is the only don't-be unit that has effects on the rest of the system. In the simulations reported here, I used $m = 4$ don't-be units per goal.

- In addition to the parameters associated with spherical units, each do-be and don't-be unit has a *strength* parameter $s$ that specifies how much influence the unit has over the behavior of the arm. Do-be strength ($s_t^+$) grows logarithmically with positive reinforcement and shrinks linearly with negative; don't-be strength ($s_{ti}^-$) grows logarithmically with negative reinforcement and shrinks linearly with positive.

Figure 5 illustrates a situation from early in a run of the system. From left to right, the three displays show the state of the relevant do-be unit, the relevant don't-be subnetwork, and the current configuration of the arm. Since this particular goal has never been achieved before, the do-be map provides no useful information — its weight vector contains small random values (as it happens, the origin is below and right of the display) and its strength is zero. The display of the don't-be map shows the positions of all four relevant don't-be units, with the currently selected don't-be (unit number 3) drawn somewhat darker. The don't-be units are spread around configuration space, creating "hills" that push away the arm if it comes too close. As the arm moves about without reaching the target, different don't-be units win the competition and take control. Negative reinforcement accrues, and the winning don't-be consequently moves toward the various current configurations and gets stronger, until the arm is pushed elsewhere.

Figure 6 illustrates the behavior of the system after more extensive learning. The

**Figure 4.** *ALVIS*

0. *(Initialize)* Given: a space $X$ of $h$ dimensions, a backpropagation network trained on $\theta \rightarrow X$, a set $G$ of goals and a mapping from $G$ to regions of space. Create an *ALVIS* network with $n = |G|$ goal units $g_1, \ldots, g_n$, $n$ do-be/don't-be subnetworks consisting of one do-be unit $d_t^+$ and $m$ don't-be units $d_{t1}^-, \ldots, d_{tm}^-$, and $h$ current position units $x_1, \ldots x_h$. Create modifiable connections $w_{xi}$ from $x$'s to $d$'s, $w_{ix}$ from $d$'s to $x$'s, and a modifiable strength $s$ for each $d$. Set all do-be strengths $s_t^+$ and all don't-be strengths $s_{ti}^-$ to zero. Set all weights $w_{xi}$ and $w_{ix}$ to small random values. Set $\theta$ to a random legal vector and produce a current configuration $X$.

1. *(New stimulus)* Choose $t$ at random from $1 \ldots n$.

2. *(Do's/Don'ts)* Compute activations for do-be's and don't-be's using the spherical function: $d = 1 / \left( 1 + \frac{1}{r} \sqrt{\sum_{i=1}^{h} (x_i - w_{xi})^2} \right)$. In subnetwork $t$, let $d_{t*}^-$ be the unit with the largest activation.

3. *(Errors)* Let $w_{ix}^+$ denote the weights from $d_t^+$ and $w_{ix}^-$ denote the weights from $d_{t*}^-$, and similarly for strengths $s_t^+$ and $s_{t*}^-$. Compute errors for each component of $X$: $e(x_i) = s_t^+ (w_{ix}^+ - x_i) + s_{t*}^- (x_i - w_{ix}^-)$.

4. *(Move)* Backpropagate to produce $e(\theta)$. Generate angle changes: $\Delta\theta_i = \min(q, \max(-q, k_r e(\theta_i)))$, with parameters $q$ and $k_r$. Generate new angles respecting the maximum $v_i^+$ and minimum $v_i^-$ possible joint angles: $\theta_i' = \min(v_i^+, \max(v_i^-, \theta_i + \Delta\theta_i))$. Forward propagate to produce a new configuration $X'$.

5. *(Positive reinforcement)* Determine whether $X'$ satisfies goal $t$. If it does not, go to step 6. Otherwise,
   5.1 For $i = 1, \ldots, h$, let $w_{ix}^{+\prime} = x_i'$ and $w_{xi}^{+\prime} = x_i'$.
   5.2 Let $s_t^{+\prime} = \min \left( 5, s_t^+ + p^+ / \left( 1 + s_t^+ \right) \right)$, for positive reinf $p^+ > 0$.
   5.3 Let $r_t^+ = k_s / \max(.1, s_t^{+\prime})$, with parameter $k_s$.
   5.4 For $i = 1, \ldots, m$, let $s_{ti}^{-\prime} = \max(0, s_{ti}^- - p^+)$, and $r_{ti}^- = k_s / \max(.1, s_{ti}^{-\prime})$.
   5.5 Go to step 1.

6. *(Negative reinforcement)* Perform the following:
   6.1 For $i = 1, \ldots, h$, let $w_{ix}^{-\prime} = w_{ix}^- + \eta(x_i' - w_{ix}^-)$ and $w_{xi}^{-\prime} = w_{xi}^- + \xi + \eta(x_i' - w_{xi}^+)$ with parameter $\eta$, where $\xi$ is a uniform random variable between $\pm 0.01$.
   6.2 Let $s_t^{-\prime} = \min \left( 5, s_t^- + p^- / \left( 1 + s_t^- \right) \right)$, for negative reinf $p^- > 0$.
   6.3 Let $r_t^- = k_s / \max(.1, s_t^{-\prime})$.
   6.4 Let $s_t^{+\prime} = \max(0, s_t^+ - p^-)$, and $r_t^+ = k_s / \max(0.1, s_t^{+\prime})$.
   6.5 Go to step 2.

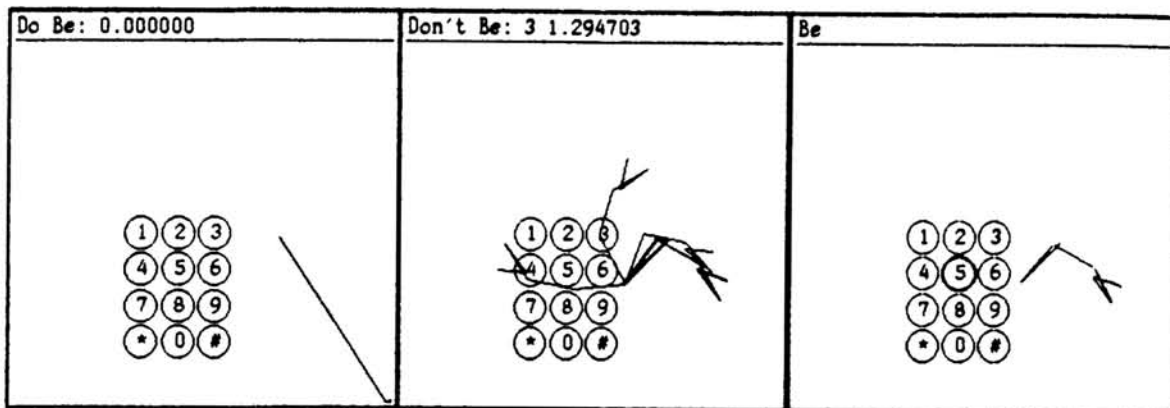

**Figure 5.** A display of the internal state of the trunk robot in the process of learning to associate a set of twelve arbitrary stimuli with specified positions in space. The current signal (though the system has not discovered this yet) means "touch 5".

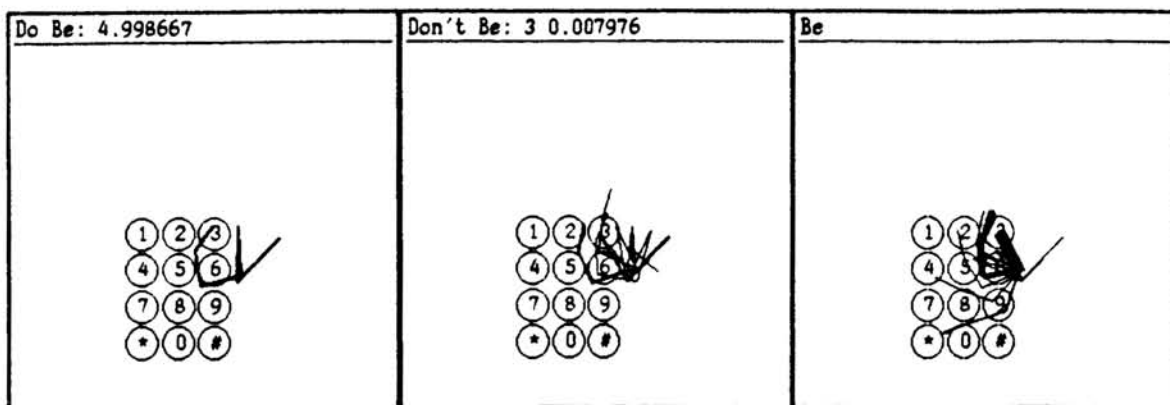

**Figure 6.** A display of the internal state of the same trunk robot later in the learning process. The previous goal was "touch *" and the current goal is "touch 3."

do–be map now contains an accurate image of a successful configuration for the "touch 3" goal, and its strength is high. The strength of the selected don't-be unit is low. The current configuration map in Figure 6 shows each iteration of the algorithm between the time it achieved its previous goal and the time it achieved the current goal.

Finally, Figure 7 displays the average time-per-goal as a function of the number of goals achieved. For 75 repetitions, the trunk network was initialized and run until 500 goals had been achieved, and the resulting time-per-goal data was averaged to produce the graph.

The average time-per-goal declines rapidly as goals are presented, then seems to rise slightly, and then stabilizes around an average value of about 300. To have some kind of standard of comparison, albeit unsophisticated, if the joint angles are simply changed by uniform random values between $q$ and $-q$ (see Figure 4) on each iteration, the average time-per-goal is observed to be about 490.

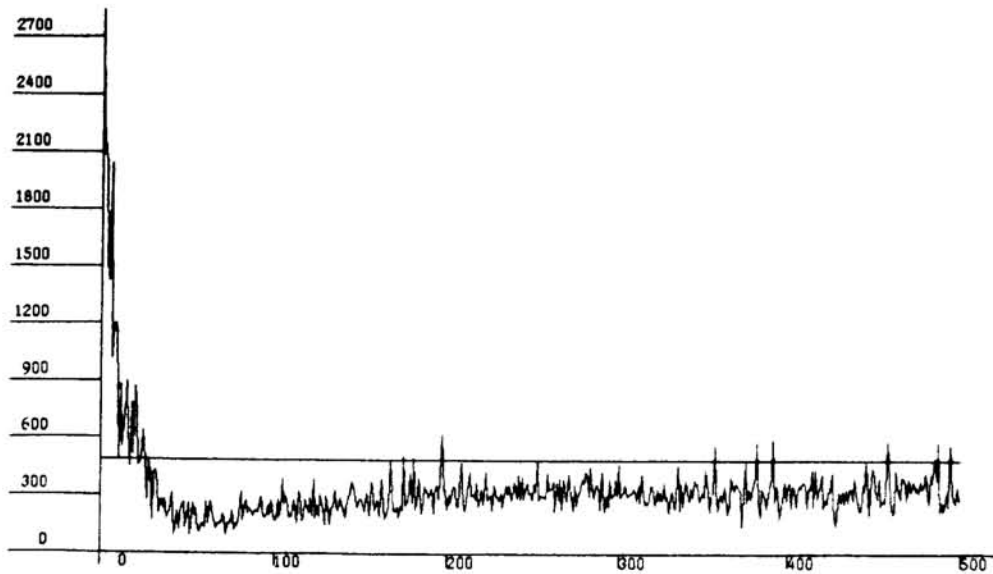

**Figure 7.** A graph of the average time taken per goal as a function of the number of goals achieved. The horizontal line shows the average performance of random joint changes.

## DISCUSSION

*ALVIS* is a preliminary, exploratory system. Of course, the *ALVIS* environment is but a pale shadow of the real world, but even granting the limited scope of the problem formulation, several aspects of *ALVIS* are incompletely satisfying and in need of improvement.

To my eye, the biggest drawback of the current implementation is the local goal representation — which essentially requires that the set of goals be enumerable at network definition time. Related problems include the inability to share information between one goal and another and the inability to pursue more than one goal simultaneously. To determine behavior, the constraints of goals must be integrated with the possibilities of the current situation. In *ALVIS* this is done as a strictly two-step process: the goal selects a subnetwork, and the current situation selects units within the subnetwork. Work such as Jordan (1986) and Miyata (1988) shows how goal information and context information can be integrated by supplying both as inputs to a single network.

*ALVIS* is a pure feedback control model, and can suffer from the traditional problem of that approach: when the errors are small, the resulting joint angle changes are small, and the arm converges only slowly. If the gain at the joints is increased to speed convergence, overshoot and oscillation become more likely. However, in *ALVIS* oscillations gradually die out, as don't-be units shift positions under the negative reinforcement, and sometimes such temporary oscillations actually help with the search, causing the tip of the arm to explore a variety of different points.

The aspect of *ALVIS* behavior that I find most irritating reveals something about the approach in general. In some cases — usually on more "peripheral" targets — *ALVIS* learns to hit the *very edge* of the target region. While approaching such targets, *ALVIS* experiences negative reinforcement, and the don't-be units, consequently, gain a little strength. The resulting interference occasionally causes a very long search for a goal that had previously been rapidly achieved. *ALVIS* has a representation only for its own body; a better system would also be able to represent other objects in the world, and useful relations on the expanded set. The "mostly motor" emphasis evident in the present system needs to be balanced by more sophistication on the perceptual side.

Though limited in scope, *ALVIS* demonstrates three ideas I think worth highlighting:

- The reuse of the error channel of a backpropagation network, after training, for translating arbitrary output-space gradients into input-space gradients.

- The recording of previous actual outputs to be used as future desired outputs.

- The use of "repellors" (don't-be units) as well as attractors in defining errors, and the resulting process of search-by-inhibition generated by negative reinforcement.

Characterizing the behavior of a machine in terms of attractor dynamics is a familiar notion, but "repellor dynamics" seems to be largely unknown territory. Indeed, in *ALVIS* there is an ephemeral quality to the don't-be units: When all answers have been discovered, all strength accrues to the do-be's, good performances become routine, and *ALVIS* behavior is essentially attractor-based. In watching such a "grown-up" *ALVIS*, it is easy to forget how it was in the beginning, when the world was big and answers were scarce, and *ALVIS* was doing well just to discover a new mistake.

## References

Ackley, D.H. (1988). Associative learning via inhibitory search. Technical memorandum TM ARH-012509, Morristown, NJ: Bell Communications Research.

Burr, D.J., & Hanson, S.J. (1987). Knowledge representation in connectionist networks. Technical memorandum TM-ARH-008733, Morristown, NJ: Bell Communications Research.

Feldman, J.A. (1982). Dynamic connections in neural networks. *Biological Cybernetics, 36*, 193–202.

Jordan, M.I. (1986). Serial order: A parallel, distributed processing approach. Technical report ICS-8604. La Jolla, CA: University of California, Institute for Cognitive Science.

Miyata, Y. (1988). The learning and planning of actions. Unpublished doctoral dissertation in psychology, University of California San Diego.

Rumelhart, D.E., Hinton, G.E., & Williams, R.J. (1986). Learning representations by back-propagating errors. *Nature, 323*, 533–536.

Rumelhart, D.E. (personal communication, 1987). Also cited as personal communication in Miyata (1988).